# Time Warping Invariant Neural Networks

## Guo-Zheng Sun, Hsing-Hen Chen and Yee-Chun Lee

Institute for Advanced Computer Studies
and
Laboratory for Plasma Research,
University of Maryland
College Park, MD 20742

## Abstract

We proposed a model of Time Warping Invariant Neural Networks (TWINN) to handle the time warped continuous signals. Although TWINN is a simple modification of well known recurrent neural network, analysis has shown that TWINN completely removes time warping and is able to handle difficult classification problem. It is also shown that TWINN has certain advantages over the current available sequential processing schemes: Dynamic Programming(DP)[1], Hidden Markov Model(-HMM)[2], Time Delayed Neural Networks(TDNN) [3] and Neural Network Finite Automata(NNFA)[4].

We also analyzed the time continuity employed in TWINN and pointed out that this kind of structure can memorize longer input history compared with Neural Network Finite Automata (NNFA). This may help to understand the well accepted fact that for learning grammatical reference with NNFA one had to start with very short strings in training set.

The numerical example we used is a trajectory classification problem. This problem, making a feature of variable sampling rates, having internal states, continuous dynamics, heavily time-warped data and deformed phase space trajectories, is shown to be difficult to other schemes. With TWINN this problem has been learned in 100 iterations. For benchmark we also trained the exact same problem with TDNN and completely failed as expected.

## I. INTRODUCTION

In dealing with the temporal pattern classification or recognition, time warping of input signals is one of the difficult problems we often encounter. Although there are a number of schemes available to handle time warping, e.g. Dynamic Programming (DP) and Hidden Markov Model(HMM), these schemes also have their own shortcomings in certain aspects. More depressing is that, as far as we know, there are no efficient neural network schemes to handle time warping. In this paper we proposed a model of Time Warping Invariant Neural Networks (TWINN) as a solution. Although TWINN is only a simple modification to the well known neural net structure, analysis shows that TWINN has the built-in ability to remove time warping completely.

The basic idea of TWINN is straightforward. If one plots the state trajectories of a continuous

dynamical system in its phase space, these trajectory curves are independent of time warping because time warping can only change the time duration when traveling along these trajectories and does not affect their shapes and structures. Therefore, if we normalize the time dependence of the state variables with respect to any phase space variable, say the length of trajectory, the neural network dynamics becomes time warping invariant.

To illustrate the power of the TWINN we tested it with a numerical example of trajectory classification. This problem, chosen as a typical problem that the TWINN could handle, has the following properties: (1). The input signals obey a continuous time dynamics and are sampled with various sampling rates. (2). The dynamics of the de-warped signals has internal states. (3). The temporal patterns consist of severely time warped signals.

To our knowledge there have not been any neural network schemes which can deal with this case effectively. We tested it with TDNN and failed to learn.

In the next section we will introduce the TWINN and prove its time warping invariance. In Section III we analyze its features and identify the advantages over other schemes. The numerical example of the trajectory classification with TWINN is presented in Section IV.

## II. TIME WARPING INVARIANT NEURAL NETWORKS (TWINN)

To process temporal signals, we consider a fully recurrent network, which consists of two groups of neurons: the state neurons (or recurrent units) represented by vector $S(t)$ and the input neurons that are clamped to the external input signals $\{I(t), t = 0, 1, 2,......, T-1\}$. The Time Warping Invariant Neural Networks (TWINN) is simply defined as:

$$S(t+1) = S(t) + l(t) F(S(t), W, I(t))$$ (1)

where $W$ is the weight matrix, $l(t)$ is the distance between two consecutive input vectors defined by the norm

$$l(t) = \| I(t+1) - I(t) \|$$ (2)

and the mapping function $F$ is a nonlinear function usually referred as neural activity function. For example of first order networks, it could take the form:

$$F_i(S(t), W, I(t)) = Tanh\left(\sum_j W_{ij}(S(t) \oplus I(t))_j\right)$$ (3)

where $Tanh(x)$ is Hyperbolic Tangent function and symbol $\oplus$ stands for the vector concatenation.

For the purpose of classification (or recognition), we assign the target final state $S_k$, (k=1,2,3,...K), for each category of patterns. After we feed into the TWINN the whole sequence $\{I(0), I(1), I(2),.......,I(T-1)\}$, the state vector $S(t)$ will reach the final state $S(T)$. We then need to compare $S(T)$ with the target final state $S_k$ for each category k, (k=1,2,3,...K), and calculate the error:

$$e_k = \| S(T) - S_k \|^2$$ (4)

The one with minimal error will be classified as such. The ideal error is zero.

For the purpose of training, we are given a set of training examples for each category. We then minimize the error functions given by Eq. (4) using either back-propagation[7] or forward propagation algorithm[8]. The training process can be terminated when the total error reach its minimum.

The formula of TWINN as shown in Eq. (1) does not look like new. The subtle difference from wildly used models is the introduction of normalization factor $l(t)$ as in Eq. (1). The main advantage by doing this lies in its built-in time warping ability. This can be directly seen from its continuous version.

As Eq. (1) is the discrete implementation of continuous dynamics, we can easily convert it into a continuous version by replacing "$t+1$" by "$t+\Delta t$" and let $\Delta t \to 0$. By doing so, we get

$$\lim_{\Delta t \to 0} \frac{S(t + \Delta t) - S(t)}{\| I(t + \Delta t) - I(t) \|} = \frac{dS}{dL} \tag{5}$$

where $L$ is the input trajectory length, which can be expressed as an integral

$$L(t) = \int_0^t \left\| \frac{dI}{d\tau} \right\| d\tau \tag{6}$$

or summation (as in discrete version)

$$L(t) = \sum_{\tau=0}^t \| I(\tau + 1) - I(\tau) \| \tag{7}$$

For deterministic dynamics, the distance $L(t)$ is a single-valued function. Therefore, we can make a unique mapping from $t$ to $L$, $\Pi: t \to L$, and any function of $t$ can be transformed into a function of $L$ in terms of this mapping. For instance, the input trajectory $I(t)$ and the state trajectory $S(t)$ can be transformed into $I(L)$ and $S(L)$. By doing so, discrete dynamics of Eq. (1) becomes, in the continuous limit,

$$\frac{dS}{dL} = F(S(L), W, I(L)) \tag{8}$$

It is obvious that there is no explicit time dependence in Eq. (8) and therefore the dynamics represented by Eq. (8) is time warping independent.

To be more specific, if we draw the trajectory curves of $I(t)$ and $S(t)$ in their phase spaces respectively, these two curves would not be deformed if we only change the time duration when traveling along the curves. Therefore, if we generate several input sequences $\{I(t)\}$ using different time warping functions and feed them into TWINN, represented by Eq. (8) or Eq. (1), the induced state dynamics of $S(L)$ would be the same. Meanwhile, the final state is the solo criterion for classification. Therefore, any time warped signals would be classified by the TWINN as the same. This is the so called "time warping invariant".

## III. ANALYSIS OF TWINN VS. OTHER SCHEMES

We emphasize two points in this section. First, we would analyze the advantages of the TWINN over the other neural network structures, like TDNN, and other mature and well known algorithms for time warping, such as HMM and Dynamics Programming. Second, we would analyze the memory capacity of input history for both the continuous dynamical networks as illustrated in Eq. (1) and its discrete companion, Neural Network Finite Automata used in grammatical inference by Liu [3], Sun [4] and Giles [5]. And, we will show by mathematical estimation that the continuity employed in TWINN increases the power of memorizing history compared with NNFA

The Time Delayed Neural Networks (TDNN)[3] has been a useful neural network structure in processing temporal signals and achieves successes in several applications, e.g. speech recognition. The traditional neural network structures are either feedforward or recurrent. The TDNN is something in between. The power of TDNN is in its dynamic combination of the spatial processing (as in a feedforward net) and sequential processing (as in a recurrent net with short time memory). Therefore, the TDNN could detect the local features within each windowed frame and store their voting scores into the short time memory neurons, and then make a final decision at the end of input sequence. This technique is suitable for processing the temporal patterns where the classification is decided by the integration of local features. But, it could not handle the long time correlation across time frames like a state machine. It also does not tolerate time warping effectively. Each of time warped patterns will be treated as a new feature. Therefore, TDNN would not be able to handle the numerical example given in this paper which has both the severe time warping and the internal states (long time correlation). The benchmark test has been performed and it proved our prediction. Actually, it can be seen later that in our exam-

ples, no matter which category they belong to, all windowed frames would contain similar local features, the simple integration of local features do not contribute directly to the final classification, rather the whole sinal history will decide the classification.

As for the Dynamic Programming, it is to date the most efficient way to cope with time warping problem. The most impressing feature of dynamic programming is that it accomplishes a global search among all $N^N$ possible paths using only $\sim O(N^2)$ operations, where N is the length of the input time series and, of course, one operation here represents all calculations involved in evaluating the 'score" of one path. But, on the other hand this is not ideal. If we can do the time warping using recurrent network, the number of operations will be reduced to $\sim O(N)$. This is a dramatic saving. Another undesirable feature of current dynamic warping scheme is that the recognition or classification result heavily depends on the pre-selected template and therefore one may need a large number of templates for a better classification rate. By adding one or two template we actually double or triple the number of operations. Therefore, search for a neural network time warping scheme is a pressing task.

Another available technique for time warping is Hidden Markov Model (HMM), which has been successfully applied in speech recognition. The way for HMM to deal with time warping is in terms of statistical behavior of its hidden state transition. Starting from one state $q_i$, HMM allows a certain probability $a_{ij}$ to forward to another state $q_j$. Therefore, for any given HMM one could generate various state sequences, say, $q_1q_2q_2q_3q_4q_4q_5$, $q_1q_2q_2q_2q_3q_3q_4q_4q_5$, etc., each with a certain occurrence probability. But, these state sequences are "hidden", the observed part is a set of speech data or symbol represented by $\{s_k\}$ for example. HMM also includes a set of observation probability $B\equiv\{b_{jk}\}$, so that when it is in a certain state, say $q_j$, HMM allows each symbol from the set $\{s_k\}$ to occur with the probability $b_{jk}$. Therefore, for any state sequence one can generate various series of symbols. As an example, let us consider one simple way to generate symbols: in state $q_j$ we generate symbol $s_j$ (with probability $b_{jj}$). By doing so, the two state sequences mentioned above would correspond to two possible symbol sequences: $s_1s_2s_2s_3s_4s_4s_5$ and $s_1s_2s_2s_2s_3s_3s_4s_4s_5$. Examining the two strings closely, we find that the second one may be considered as the time warped version of the first one, or *vice versa*. If we present these two strings to the HMM for testing, it will accept them with similar probabilities. This is the way that HMM tolerates time warping. And, these state transition probabilities of HMM are learned from the statistics of training set by using re-estimation formula. In this sense, HMM does not deal with time warping directly, instead, it learns statistical distribution of training set which contains time warped patterns. Consequently, if one presents a test pattern with time warped signals which is far away from the statistical distribution of training set, it is very unlikely for a HMM to recognize this pattern.

On the contrary, the model of TWINN we proposed here has intrinsic built-in time warping nature. Although the TWINN itself has internal states, these internal states are not used for tolerating time warping. Instead, they are used to learn more complex behavior of the "de-warped" trajectories. In this sense, TWINN could be more powerful than HMM.

Another feature of TWINN needs be mention is its explicit expression of continuous mapping from $S(t)$ to $S(t+1)$ as shown in Eq. (1). In our early work of [4,5,6], to train a NNFA (Neural Network Finite Automaton), we used a discrete mapping

$$S(t+1) = F(S(t), W, I(t)) \tag{9}$$

where $F$ is a nonlinear function, say *Sigmoid* function $g(x) \equiv 1/(1+e^{-x})$. This model has been successfully applied into the grammatical inference. The reason we call Eq. (1) a continuous mapping but Eq. (9) a discrete one, even though both of them are implemented in discrete time steps, is because there is an explicit infinitesimal factor $l(t)$ used in Eq. (1). Due to this factor the continuous state dynamics is guaranteed, by which we mean that the state variation $S(t+1)$ - $S(t+1)$ approaches zero if the input variation $I(t+1)$ - $I(t+1)$ does so. But, In general, the state

variation $S(t+1) - S(t+1)$ generated by Eq. (9) is of order of one, regardless of what input variations are. If one starts from random initial weights, Eq. (9) provides a discrete jump between different, randomly distributed states, which is far away from any continuous dynamics.

We did numerical test using NNFA of Eq. (9) to learn the classification problem of continuous trajectories as shown in Section V. For simplicity we did not include time warping, but the NNFA still failed to learn. The reason is that when we tried to train a NNFA to learning the continuous dynamics, we were actually forcing the weights to generate an almost identical mapping $F$ from $S(t)$ to $S(t+1)$. This is a very strong constrain on the weight parameters, such that it drives the diagonal terms to positive infinity and off-diagonal terms to negative infinity (*Sigmoid* function is used). When this happens, the learning is stuck due to the saturation effect.

The failure of NNFA may also comes from the short history memory capacity compared to the continuous mapping of Eq. (1). It has been shown by many numerical experiments on grammatical inference [3, 4, 5] that to train an NNFA as in Eq. (9) effectively, one has to start with short training patterns (usually, the sentence length $\leq 4$). Otherwise, learning will fail or be very slow. This is exactly what happened to learning the trajectory classification using NNFA, where the lengths of our training patterns are in general considerably long (normally,~ 60). But, TWINN learned it easily. To understand the NNFA's failure and TWINN's success, in the following, we will analyze how the history information enters the learning process.

Consider the example of learning grammatical inference. Before training since we have no *a priori* knowledge about the target values of weights, we normally start with random initial values. On the other hand, during training the credit assignment (or the weight correction $\Delta W$) can only be done at the end of each input sequence. Consequently, each $\Delta W$ should explicitly contain the information about all symbols contained in that string, otherwise the learning is meaningless. But, in numerical implementation, every variable, including both $\Delta W$ and $W$, has a finite precision and any information beyond the precision range will be lost. Therefore, to compare which model has the longer history memory we need to examine how the history information relates to the finite precisions of $\Delta W$ and $W$.

Let us illustrate this point with a simple second-order connected fully recurrent network and write both Eq. (1) and Eq. (9) in a unified form

$$S(t+1) = G^{t+1} \tag{10}$$

such that Eq. (1) is represented by

$$G^{t+1} = S(t) + I(t) g(K(t)) \tag{11}$$

and Eq. (9) is just

$$G^{t+1} = g(K(t)) \tag{12}$$

where $K(t)$ is the weighted sum of concatenation of vectors $S(t)$ and $I(t)$

$$K_i(t) = \sum_j W_{ij}(S(t) \oplus I(t))_j \tag{13}$$

For a grammatical inference problem the error is calculated from the final state $S(T)$ as

$$E = (S(T) - S_{target})^2 \tag{14}$$

Learning is to minimize this error function. According to the standard error back-propagation scheme, the recurrent net can be viewed as a multi-layered net with identical weights between neurons at adjacent time step: $w(t) = W$, where $w(t)$ is the "$t_{th}$ layer" weights connecting *input* $S(t-1)$ to *output* $S(t)$. The total weight correction is the summation of all weight corrections at each layer. By using the gradient descent scheme one immediately has

$$\Delta W = \sum_{t=1}^{T} \delta w(t) = -\eta \sum_{t=1}^{T} \frac{\partial E}{\partial w(t)} = -\eta \sum_{t=1}^{T} \frac{\partial E}{\partial S(t)} \cdot \frac{\partial G^t}{\partial w(t)} \tag{15}$$

If we define new symbols: vector $u(t)$, second-order tensor $A(t)$ and third-order tensor $B(t)$ as

$$u_i(t) \equiv \frac{\partial E}{\partial S_i(t)} \qquad B_{ijk}(t) \equiv \frac{\partial G_i^l}{\partial W_{jk}} \qquad A_{ij}(t) \equiv \frac{\partial G_i^{l+1}}{\partial S_j(t)} \qquad (16)$$

the weight correction can be simply written as

$$\Delta W = -\eta \sum_{t=1}^{T} u(t) \cdot B(t) \qquad (17)$$

and the "error rate" $u(t)$ can be back-propagated using the Derivative Chain Rule

$$u(t) = u(t+1) \cdot A(t) \qquad t = 1, 2, ..., T-1; \qquad (18)$$

so that it is easy to have

$$u(t) = u(T) \cdot A(T-1) \cdot A(T-2) \cdot ... \cdot A(t) \equiv u(T) \cdot \prod_{\tau=T-1}^{t} A(\tau) \qquad t = 1, 2, ..., T-1; \qquad (19)$$

First, let us examine the model of NNFA in Eq. (9). Using Eqs. (12), (13) and (16), $A_{ij}(t)$ and $B_{ijk}(t)$ can be written as

$$A_{ij}(t) = g'(K_i(t)) W_{ij} \qquad B_{ijk}(t) = \delta_{ij} (S(t-1) \oplus I(t-1))_k \qquad (20)$$

where $g'(x) \equiv dg/dx = g(1-g)$ is the derivative of *Sigmoid* function and $\delta_{ij}$ is Kronecker delta function. If we substitute $B_{ijk}(t)$ into Eq. (17), $\Delta W$ becomes a weighted sum of all input symbols $\{I(0), I(1), I(2),.......,I(T-1)\}$, each with different weighting factor $u(t)$. Therefore, to guarantee that $\Delta W$ contain the information of all input symbols $\{I(0), I(1), I(2),......,I(T-1)\}$, the ratio of $|u(t)|_{max}/|u(t)|_{min}$ should be within the range of precision of $\Delta W$. This is the main point.

The exact mathematical analysis has not been done, but from a rough estimate we can gain some good understanding. From Eq. (19), $u(t)$ is a matrices product of $A_{ij}(t)$, and $u(1)$ the coefficient of $I(0)$ contains the highest order product of $A_{ij}(t)$. The key point is that the coefficient ratio between the adjacent symbols: $|u(t)|/|u(t+1)|$ is of the order of $|A_{ij}(t)|$, which is a small value, therefore the earlier symbol information could be lost from $\Delta W$ due to its finite precision. It can be shown that $xg'(x) = x g(x)(1-g(x)) < 0.25$ for any real value of x. Then, we roughly have $|A_{ij}(t)| = |g' W_{ij}| = |g(1-g)W_{ij}| < 0.25$, if we assume the values of weights $W_{ij}$ to be order 1. Thus, the ratio $R=|u(t)|_{max}/|u(t)|_{min}$ is estimated as

$$R \sim |u(1)|/|u(T)| \sim \prod_{t'=T-1}^{1} |A(t')| < 2^{-2 \cdot (T-1)} \qquad (21)$$

From Eq. (21) we see that if the input pattern length is $T=10$ we need at least $2(T-1) \cong 18$ bits computer memory to store weight variables (including $u$, W and $\Delta W$). If $T=60$, as in the trajectory classification problem, it requires at least 128 bit weight variables. This is why the NNFA Eq. (9) could not work.

Similarly, for the dynamics of Eq. (1), we use Eqs. (11), (13) and (16), and obtain

$$A_{ij}(t) = 1 + I(t) (g'(K_i(t)) W_{ij}), \qquad B_{ijk}(t) = I(t) (\delta_{ij} (S(t-1) \oplus I(t-1))_k) \qquad (22)$$

From Eq. (22) we see that no matter how small the factor l(t) will be, $|A_{ij}(t)|$ remains a value of order of one, therefore the ratio $R=|u(t)|_{max}/|u(t)|_{min}$ which is estimated as a product of $|A_{ij}(t)|$ would be of order of one compared with result of discrete case as in Eq. (21).Therefore, the contributions from *all* $\{I(0), I(1), I(2),.......,I(T-1)\}$ to the weight correction $\Delta W$ are of the same order. This prevents the information loss during learning.

## IV NUMERICAL SIMULATION

We demonstrate the power of TWINN with a trajectory classification problem. The three 2-

D trajectory equations are artificially given by

$$\begin{cases} x(t) = \sin(t+\beta)\,|\sin(t)| \\ y(t) = \cos(t+\beta)\,|\sin(t)| \end{cases} \begin{cases} x(t) = \sin(0.5t+\beta)\sin(1.5t) \\ y(t) = \cos(0.5t+\beta)\sin(1.5t) \end{cases} \begin{cases} x(t) = \sin(t+\beta)\sin(2t) \\ y(t) = \cos(t+\beta)\sin(2t) \end{cases} \quad (23)$$

where $\beta$ is a uniformly distributed random parameter. When $\beta$ is changed, these trajectories are distorted accordingly. Some examples (three for each class) are shown in Fig.1.

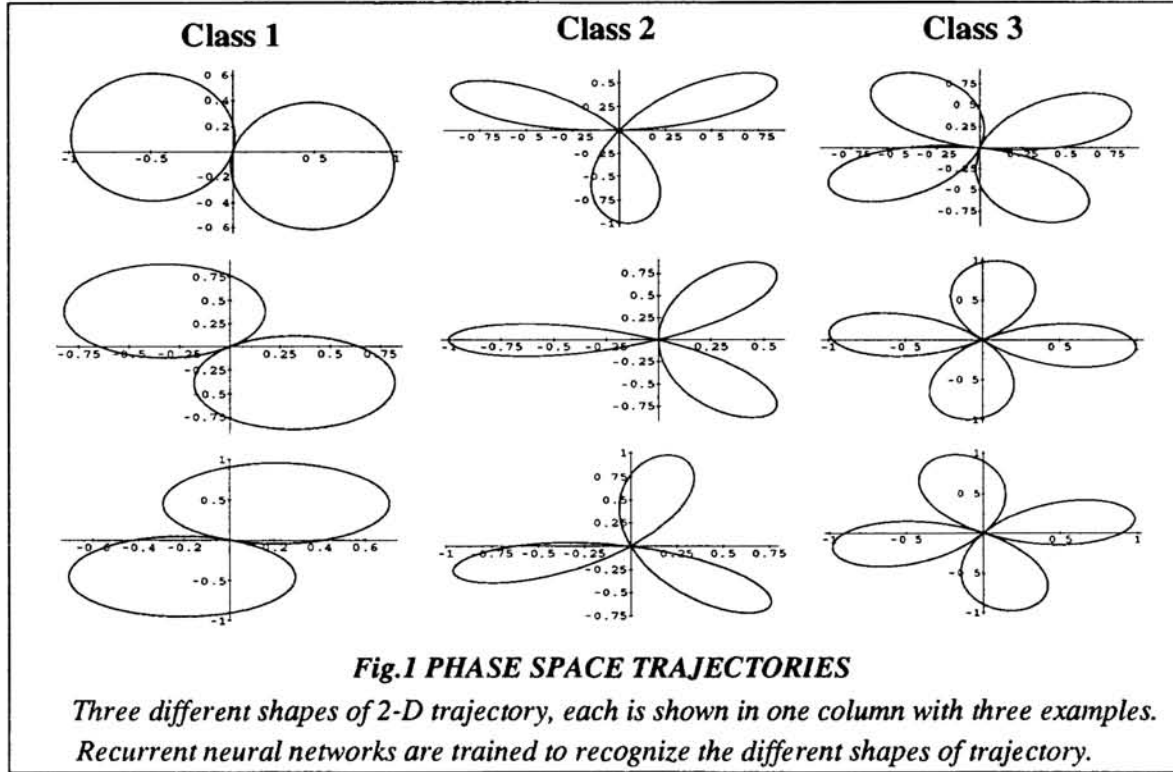

**Fig.1 PHASE SPACE TRAJECTORIES**

*Three different shapes of 2-D trajectory, each is shown in one column with three examples.*
*Recurrent neural networks are trained to recognize the different shapes of trajectory.*

The trajectory data are the time series of two dimensional coordinate pairs $\{x(t), y(t)\}$ sampled along three different types of curves in the phase space. The neural net dynamics of TWINN is

$$S_i(t+1) = S_i(t) + l(t)\left(Tanh\left(\sum_{j=1}^{N+6} W_{ij}(S(t) \oplus I(t))_j\right)\right) \quad and \quad l(t) = \sqrt{\sum_{i=1}^{6}(I_i(t) - I_i(t-1))^2} \quad (24)$$

where we used 6 input neurons $I = \{1, x(t), y(t), x^2(t), y^2(t), x(t)y(t)\}$ (normalized to norm = 1.0) and 4 ($N$=4) state neurons $S = \{S_1, S_2, S_3, S_4\}$. The neural network structure is shown in Fig. 2.

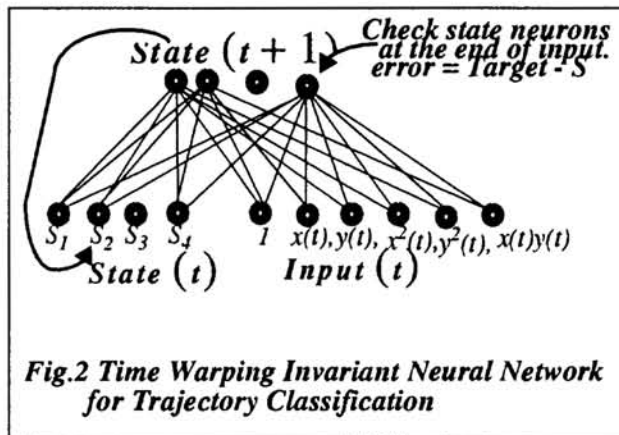

**Fig.2 Time Warping Invariant Neural Network for Trajectory Classification**

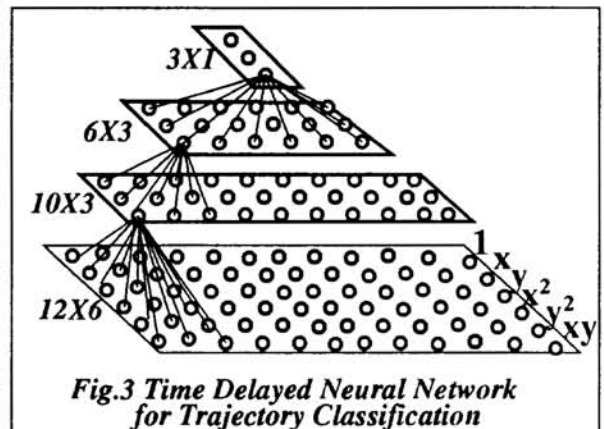

**Fig.3 Time Delayed Neural Network for Trajectory Classification**

For training, we assign the desired final output for the three trajectory classes to be (1,0,0),

(0,1,0) and (0,0,1) respectively. For recognition, each trajectory data sequence needs to be fed to the input neurons and the state neurons evolve according to the dynamics in Eq. (24). At the end of input series we check the last three state neurons and classify the input trajectory according to the "winner-take-all" rule.

In each iteration of training we randomly picked up 150 deformed trajectories, 50 for each of the three categories, by choosing different values of $\beta$ within $0 \leq \beta \leq 2\pi$. To simulate time warping we randomly sampled the data by choosing the random time step $\Delta t = 2\pi r/T$ along each trajectory, where r is a random number between 0 and 2 and the sampling rate T=60 for training patterns, and T=20 to 200 for testing patterns. Therefore, each training pattern is a time warped trajectory data with averaged length = 60. Using RTRL algorithm[8] to minimize the error function, after 100 iterations of training it converged to Mean Square Error of $\cong 0.03$.

We tested the trained network with hundreds of randomly picked input sequences with different sampling rate (from $20/2\pi$ to $200/2\pi$) and different wrapping functions (non-uniform step length). All input trajectories are classified correctly. If the sampling rates are too large (>200) or too small(<20), some classification errors will occur.

We test the same example with TDNN. See Fig.3 for its parameters. The top layer contains three output neurons for the three classes of trajectories. The classification rules, error function and training patterns are the same as those of TWINN. After three days of training with DEC-3100 Workstation the training error (MSE) approaches 0.5 and in testing the error rate is 70%.

## V. CONCLUSION

We have proposed a model of Time Warping Invariant Neural Network to handle temporal pattern classification where the severely time warped and deformed data may occur. This model is shown to have built-in time warping ability. We have analyzed the properties of TWINN and shown that for trajectory classification it has several advantages over other schemes: HMM, DP, TDNN and NNFA.

We also numerically implemented the TWINN and trained a trajectory classification easily. This problem is shown by analysis to be difficult to other schemes. It has been trained with TDNN but failed.

## References

[1] H.Sakoe and S. Chiba, "Dynamic Programming Algorithm Optimization for Spoken Word Recognition", IEEE Transactions on Acoustics Speech and Signal Processing, Vol. ASSP-26, pp.43-49, Feb. 1978.

[2] L.R.Rabiner and B.H.Juang, "An Introduction to Hidden Markov Models", IEEE, ASSP Mag., Vol.3, No. 1, pp. 4-16, 1986.

[3]A. Weibel, T. Hanazawa, G. Hinton, K.shikano and K. Lang, "Phoneme Recognition Using Time-Delay Neural Networks", IEEE Transactions on Acoustics Speech and Signal Processing, March,1989.

[4]. Y.D. Liu, G.Z. Sun, H.H. Chen, C.L. Giles and Y.C. Lee, *"Grammatic Inference and Neural Network State Machine"*, Proceedings of the International Joint Conference on Neural Networks, pp. I-285, Washington D.C. (1990).

[5]. G.Z. Sun, H.H. Chen, C.L. Giles, Y.C. Lee and D. Chen, *"Connectionist Pushdown Automata that Learn Context-Free Grammars"*, Proceedings of the International Joint Conference on Neural networks, pp. I-577, Washington D.C. (1990).

[6]Giles, C.L., Sun, G.Z., Chen, H.H., Lee,Y.C., and Chen, D. (1990). "Higher Order Recurrent Networks & Grammatical Inference". *Advances in Neural Information Processing Systems* 2, D.S. Touretzky (editor), 380-386, Morgan Kaufmann, San Mateo, C.A. (7)

[7] D.Rumelhart, G. Hinton, and R. Williams. "Learning internal representations by error propagation", In PDP: Vol.I MIT press 1986. P. Werbos, "Beyond Regression: New tools for prediction and analysis in the behavior sciences", Ph.D. thesis, Harvard university, 1974.

[8] R. Williams and D. Zipser, "A learning algorithm for continually running fully recurrent neural networks", Neural Computation 1(1989), pp.270-280.
